# Model selection and velocity estimation using novel priors for motion patterns

**Shuang Wu**
Department of Statistics
UCLA, Los Angeles, CA 90095
shuangw@stat.ucla.edu

Hongjing Lu
Department of Psychology
UCLA, Los Angeles, CA 90095
hongjing@ucla.edu

Alan Yuille
Department of Statistics
UCLA
Los Angeles, CA 90095
yuille@stat.ucla.edu

## Abstract

Psychophysical experiments show that humans are better at perceiving rotation and expansion than translation. These findings are inconsistent with standard models of motion integration which predict best performance for translation [6]. To explain this discrepancy, our theory formulates motion perception at two levels of inference: we first perform model selection between the competing models (e.g. translation, rotation, and expansion) and then estimate the velocity using the selected model. We define novel prior models for smooth rotation and expansion using techniques similar to those in the slow-and-smooth model [17] (e.g. Green functions of differential operators). The theory gives good agreement with the trends observed in human experiments.

## 1  Introduction

As an observer moves through the environment, the retinal image changes over time to create multiple complex motion flows, including translational, circular and radial motion. Human observers are able to process different motion patterns and infer ego motion and global structure of the world. However, the inherent ambiguity of local motion signals requires the visual system to employ an efficient integration strategy to combine many local measurements in order to perceive global motion. Psychophysical experiments have identified a variety of phenomena, such as motion capture and motion cooperativity [11], which appear to be consequences of such integration. A number of computational Bayesian models have been proposed to explain these effects based on prior assumptions about motion. In particular, it has been shown that a slow-and-smooth prior, and related models, can qualitatively account for a range of experimental results [17, 15, 16] and can quantitatively account for others [7, 12].

However, the integration strategy modeled by the slow-and-smooth prior may not generalize to more complex motion types, such as circular and radial motion, which are critically important for estimating ego motion. In this paper we are concerned with two questions. (1) What integration priors should be used for a particular motion input? (2) How can local motion measurements be combined with the proper priors to estimate motion flow? Within the framework of Bayesian inference, the answers to these two questions are respectively based on model selection and parameter estimation. In the field of motion perception, most work has focused on the second question, using parameter estimation to estimate motion flow. However, Stocker and Simoncelli [13] recently proposed a conditioned Bayesian model in which strong biases in precise motion direction estimates arise as a consequence of a preceding decision about a particular hypothesis (left vs. right motion).

The goal of this paper is to provide a computational explanation for both of the above questions using Bayesian inference. To address the first question, we develop new prior models for smooth rotation and expansion motion. To address the second, we propose that the human visual system has available multiple models of motion integration appropriate for different motion patterns. The visual system decides the best integration strategy based upon the perceived motion information, and this choice in turn affects the estimation of motion flow.

In this paper, we first present a computational theory in section (3) that includes three different integration strategies, all derived within the same framework. We test this theory in sections (4,5) by comparing its predictions with human performance in psychophysical experiments, in which subjects were asked to discriminate motion direction in translational, rotational, and expanding stimuli. We employ two commonly used stimuli, random dot patterns and moving gratings, to show that the model can apply to a variety of inputs.

## 2   Background

There is an enormous literature on visual motion phenomena and there is only room to summarize the work most relevant to this paper. Our computational model relates most closely to work [17, 15, 7] that formulates motion perception as Bayesian inference with a prior probability biasing towards slow-and-smooth motion. But psychophysical [4, 8, 1, 6], physiological [14, 3] and fMRI data [9] suggests that humans are sensitive to a variety of motion patterns including translation, rotation, and expansion. In particular, Lee *et al* [6] demonstrated that human performance on discrimination tasks for translation, rotation, and expansion motion was inconsistent with the predictions of the slow-and-smooth theory (our simulations independently verify this result). Instead, we propose that human motion perception is performed at two levels of inference: (i) model selection, and (ii) estimating the velocity with the selected model. The concept of model selection has been described in the literature, see [5], but has only recently been applied to model motion phenomena [13]. Our new motion models for rotation and expansion are formulated very similarly to the original slow-and-smooth model [17] and similar mathematical analysis [2] is used to obtain the forms of the solutions in terms of Greens functions of the differential operators used in the priors.

## 3   Model Formulation

### 3.1   Bayesian Framework

We formulate motion perception as a problem of Bayesian inference with two parts. The first part selects a model that best explains the observed motion pattern. The second part estimates motion properties using the selected model.

The velocity field $\{\vec{v}\}$ is estimated from velocity measurements $\{\vec{u}\}$ at discrete positions $\{\vec{r}_i, \ i = 1, \ldots N\}$ by maximizing

$$p(\{\vec{v}\}|\{\vec{u}\}, M) = \frac{p(\{\vec{u}\}|\{\vec{v}\})p(\{\vec{v}\}|M)}{p(\{\vec{u}\}|M)}, \tag{1}$$

The prior

$$p(\{\vec{v}\}|M) = \exp(-E(\{\vec{v}\}|M)/T), \tag{2}$$

differs for different models $M$ and is discussed in section 3.2.

The likelihood function

$$p(\{\vec{u}\}|\{\vec{v}\}) = \exp(-E(\{\vec{u}\}|\{\vec{v}\})/T) \tag{3}$$

depends on the measurement process and is discussed in section 3.3.

The best model that explains measurement $\{\vec{u}\}$ is chosen by maximizing the model evidence

$$p(\{\vec{u}\}|M) = \int p(\{\vec{u}\}|\{\vec{v}\})p(\{\vec{v}\}|M)d\{\vec{v}\} \tag{4}$$

which is equivalent to maximizing the posterior probability of the model $M$ (assuming uniform prior on the models):

$$M^* = \arg\max_M P(M|\{\vec{u}\}) = \arg\max_M \frac{P(\{\vec{u}\}|M)P(M)}{P(\{\vec{u}\})} = \arg\max_M P(\{\vec{u}\}|M). \tag{5}$$

## 3.2  The Priors

We define three priors corresponding to the three different types of motion – translation, rotation, and expansion. For each motion type, we encourage slowness and smoothness. The prior for translation is very similar to the slow-and-smooth prior [17] except we drop the higher-order derivative terms and introduce an extra parameter (to ensure that all three models have similar degrees of freedom).

We define the priors by their energy functions $E(\{\vec{v}\}|M)$, see equation (2). We label the models by $M \in \{t, r, e\}$, where $t, r, e$ denote translation, rotation, and expansion respectively. (We note that the prior for expansion will also account for contraction).

1. slow-and-smooth-translation:

$$E(\{\vec{v}\}|M = t) = \int \lambda(|\vec{v}|^2 + \mu|\nabla\vec{v}|^2 + \eta|\nabla^2\vec{v}|^2)d\vec{r} \qquad (6)$$

2. slow-and-smooth-rotation:

$$E(\{\vec{v}\}|M = r) = \int \lambda\{|\vec{v}|^2 + \mu[(\frac{\partial v_x}{\partial x})^2 + (\frac{\partial v_y}{\partial y})^2 + (\frac{\partial v_x}{\partial y} + \frac{\partial v_y}{\partial x})^2] + \eta|\nabla^2\vec{v}|^2\}d\vec{r} \quad (7)$$

3. slow-and-smooth-expansion:

$$E(\{\vec{v}\}|M = e) = \int \lambda\{|\vec{v}|^2 + \mu[(\frac{\partial v_x}{\partial y})^2 + (\frac{\partial v_y}{\partial x})^2 + (\frac{\partial v_x}{\partial x} - \frac{\partial v_y}{\partial y})^2] + \eta|\nabla^2\vec{v}|^2\}d\vec{r} \quad (8)$$

These models are motivated as follows. The $|\vec{v}|^2$ and $|\nabla^2\vec{v}|^2$ bias towards slowness and smoothness and are common to all models. The first derivative term gives the differences among the models. The translation model prefers constant translation motion with $\vec{v}$ constant, since $\nabla\vec{v} = 0$ for this type of motion. The rotation model prefers rigid rotation and expansion, respectively, of ideal form

$$\{v_x = -\omega(y - y_0), v_y = \omega(x - x_0)\}, \{v_x = e(x - x_0), v_y = e(y - y_0) \qquad (9)$$

where $(x_0, y_0)$ are the (unknown) centers, $\omega$ is the angular speed and $e$ is the expansion rate. These forms of motion are preferred by the two models since, for the first type of motion (rotation) we have $\{\frac{\partial v_x}{\partial y} + \frac{\partial v_y}{\partial x} = 0, \frac{\partial v_x}{\partial x} = \frac{\partial v_y}{\partial y} = 0\}$ (independent of $(x_0, y_0)$ and $\omega$). Similarly, the second type of motion is preferred by the expansion (or contraction) model since $\{\frac{\partial v_x}{\partial x} - \frac{\partial v_y}{\partial y} = 0, \frac{\partial v_x}{\partial y} = \frac{\partial v_y}{\partial x} = 0\}$ (again independent of $(x_0, y_0)$ and $e$).

The translation model is similar to the first three terms of the slow-and-smooth energy function [17] but with a restriction on the set of parameters. Formally $\lambda(|\vec{v}|^2 + \frac{\sigma^2}{2}|\nabla\vec{v}|^2 + \frac{\sigma^4}{8}|\nabla^2\vec{v}|^2)d\vec{r}$ $\approx \lambda \sum_{m=0}^{\infty} \frac{\sigma^{2m}}{m!2^m}|D^m\vec{v}|^2 d\vec{r}$. Our computer simulations showed that the translation model performs similar to the slow-and-smooth model.

## 3.3  The Likelihood Functions

The likelihood function differs for the two classes of stimuli we examined: (i) For the moving dot stimuli, as used in [4], there is enough information to estimate the local velocity $\vec{u}$; (ii) For the gratings stimuli [10], there is only enough information to estimate one component of the velocity field.

For the dot stimuli, the energy term in the likelihood function is set to be

$$E(\{\vec{u}|\vec{v}\}) = \sum_{i=1}^{N} |\vec{v}(\vec{r}_i) - \vec{u}(\vec{r}_i)|^2 \qquad (10)$$

For the gratings stimuli, see 2, the likelihood function uses the energy function

$$E_n(\{\vec{u}\}|\{\vec{v}\}) = \sum_{i=1}^{N} |\vec{v}(\vec{r}_i) \cdot \hat{\vec{u}}(\vec{r}_i) - |\vec{u}(\vec{r}_i)||^2 \qquad (11)$$

where $\hat{\vec{u}}(\vec{r}_i)$ is the unit vector in the direction of $\vec{u}(\vec{r}_i)$ and normally it is the direction of local image gradient.

### 3.4 MAP estimator of velocities

The MAP estimate of the velocities for each model is obtained by solving

$$\vec{v}^* = \arg\max_{\vec{v}} p(\{\vec{v}\}|\{\vec{u}\}, M) = \arg\min_{\vec{v}}\{E(\{\vec{u}|\vec{v}\}) + E(\{\vec{v}\}|M)\} \tag{12}$$

For the slow-and-smooth model [17], it was shown using regularization analysis [2] that this solution can be expressed in terms of a linear combination of the Green function $G$ of the differential operator which imposes the slow-and-smoothness constraint (the precise form of this constraint was chosen so that $G$ was a Gaussian).

We can obtain similar results for the three types of models $M \in \{t, r, e\}$ we have introduced in this paper. The main difference is that the models require two vector valued Green functions $\vec{G}_1^M = (G_{1x}^M, G_{1y}^M)$ and $\vec{G}_2^M = (G_{2x}^M, G_{2y}^M)$, with the constraint that $G_{1x}^M = G_{2y}^M$ and $G_{2x}^M = G_{1y}^M$. These vector-valued Green functions are required to perform the coupling between the different velocity component required for rotation and expansion, see figure (1). For the translation model there is no coupling required and so $G_{2x}^M = G_{1y}^M = 0$.

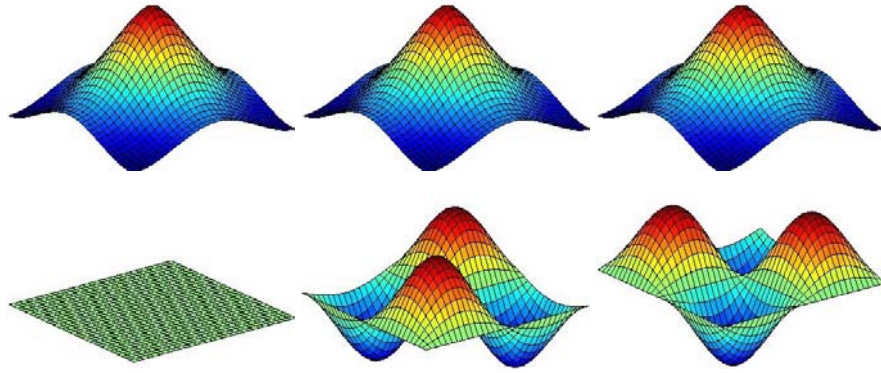

Figure 1: The vector-valued Green function $\vec{G} = (G_1, G_2)$. Top panel, left-to-right: $G_{1x}^{M=t}, G_{1x}^{M=r}, G_{1x}^{M=e}$ for the translation, rotation and expansion models. Bottom panel: left-to right: $G_{2x}^{M=t}, G_{2x}^{M=r}, G_{2x}^{M=e}$ for translation, rotation, and expansion models. Observe that the $G_{1x}^M$ are similar for all models, $G_{2x}^{M=t}$ vanishes for the translation model (i.e. no coupling between velocity components), and $G_{2x}^{M=r}$ and $G_{2x}^{M=e}$ both have two peaks which correspond to the two directions of rotation and expansion. Recall that $G_{1y}^M = G_{2x}^M$ and $G_{2y}^M = G_{1x}^M$.

The estimated velocity for the $M$ model is of the form:

$$\vec{v}(\vec{r}) = \sum_{i=1}^{N}[\alpha_i \vec{G}_1^M(\vec{r} - \vec{r}_i) + \beta_i \vec{G}_2^M(\vec{r} - \vec{r}_i)], \tag{13}$$

For the dot stimuli, the $\{\alpha\}, \{\beta\}$ are obtained by solving the linear equations:

$$\sum_{j=1}^{N}[\alpha_j \vec{G}_1^M(\vec{r}_i - \vec{r}_j) + \beta_j \vec{G}_2^M(\vec{r}_i - \vec{r}_j)] + \alpha_i \vec{e}_1 + \beta_i \vec{e}_2 = \vec{u}(r_i), \ i = 1, \dots N, \tag{14}$$

where $\vec{e}_1, \vec{e}_2$ denote the (orthogonal) coordinate axes. If we express the $\{\alpha\}, \{\beta\}$ as two N-dim vectors $A$ and $B$, the $\{u_x\}$ and $\{u_y\}$ as vectors $U = (U_x, U_y)^T$, and define $N \times N$ matrices $g_{1x}^M, g_{2x}^M, g_{1y}^M, g_{2y}^M$ to have components $G_{1x}^M(\vec{r}_i - \vec{r}_j), G_{2x}^M(\vec{r}_i - \vec{r}_j), G_{1y}^M(\vec{r}_i - \vec{r}_j), G_{2y}^M(\vec{r}_i - \vec{r}_j)$ respectively, then we can express these linear equations as:

$$\begin{pmatrix} g_{1x}^M + I & g_{2x}^M \\ g_{1y}^M & g_{2y}^M + I \end{pmatrix} \begin{pmatrix} A \\ B \end{pmatrix} = \begin{pmatrix} U_x \\ U_y \end{pmatrix} \tag{15}$$

Similarly for the gratings stimuli,

$$\begin{pmatrix} \tilde{g}_{1x}^M + I & \tilde{g}_{2x}^M \\ \tilde{g}_{1y}^M & \tilde{g}_{2y}^M + I \end{pmatrix} \begin{pmatrix} A \\ B \end{pmatrix} = \begin{pmatrix} U_x \\ U_y \end{pmatrix} \tag{16}$$

in which $\tilde{g}_{1x}^M$ is the matrix with components $\tilde{G}_{1x}^M(\vec{r}_i - \vec{r}_j) = [\vec{G}_1^M(\vec{r}_i - \vec{r}_j) \cdot \hat{\tilde{u}}(r_i)]\hat{\tilde{u}}_x(r_i)$, and similarly for $\tilde{g}_{1y}^M$, $\tilde{g}_{2x}^M$ and $\tilde{g}_{2y}^M$.

## 3.5 Model Selection

We re-express model evidence $p(\{\vec{u}\}|M)$ in terms of $(A, B)$:

$$p(\{\vec{u}\}|M) = \int p(\{\vec{u}\}|A, B, M)p(A, B)dAdB \tag{17}$$

We introduce new notation in the form of $2N \times 2N$ matrices: $g^M = \begin{pmatrix} g_{1x}^M & g_{2x}^M \\ g_{1y}^M & g_{2y}^M \end{pmatrix}$, similarly for $\tilde{g}^M$.

The model evidence for the dot stimuli can be computed analytically (exploiting properties of multi-dimensional Gaussians) to obtain:

$$p(\{\vec{u}\}|M) = \frac{1}{(\pi T)^N \sqrt{\det(g^M + I)}} \exp[-\frac{1}{T}(U^T U - U^T \frac{g^M}{g^M + I}U)] \tag{18}$$

Similarly, for the gratings stimuli we obtain:

$$p(\{\vec{u}\}|M) = \frac{1}{(\pi T)^N} \frac{\sqrt{\det(g^M)}}{\sqrt{\det(\tilde{\Sigma})}} \exp[-\frac{1}{T}(U^T U - U^T \tilde{g}^M \tilde{\Sigma}^{-1}(\tilde{g}^M)^T U)] \tag{19}$$

where $\tilde{\Sigma} = (\tilde{g}^M)^T \tilde{g}^M + g^M$.

# 4  Results on random dot motion

We first investigate motion perception with the moving dots stimuli used by Freeman and Harris [4], as shown in figure (2). The stimuli consist of 128 moving dots in a random spatial pattern. All the dots have the same speed in all three motion patterns, including translation, rotation and expansion. Our simulations first select the correct model for each stimulus and then estimate the speed threshold of detection for each type of motion. The parameter values used are $\lambda = 0.001$, $\mu = 12.5$, $\eta = 78.125$ and $T = 0.0054$.

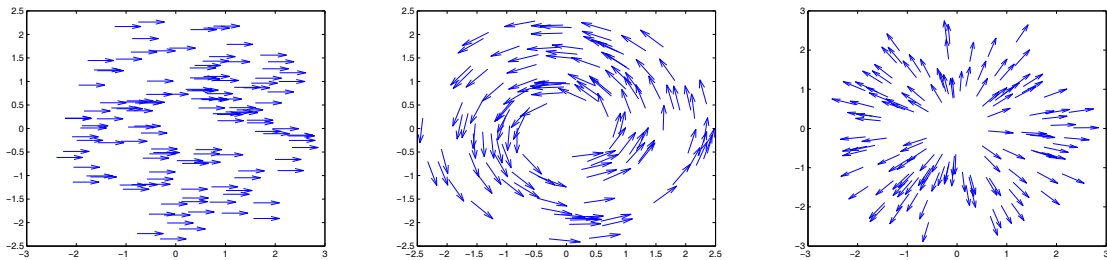

Figure 2: Moving random dot stimuli. Left panel: translation; middle panel: rotation; right panel: expansion.

## 4.1 Model selection

Model selection results are shown in figure (3). As speed increases in the range of 0.05 to 0.1, model evidence decreases for all models. This is due to slowness term in all model priors. Nevertheless the correct model is always selected over the entire range of speed, and for all 3 type of motion stimuli.

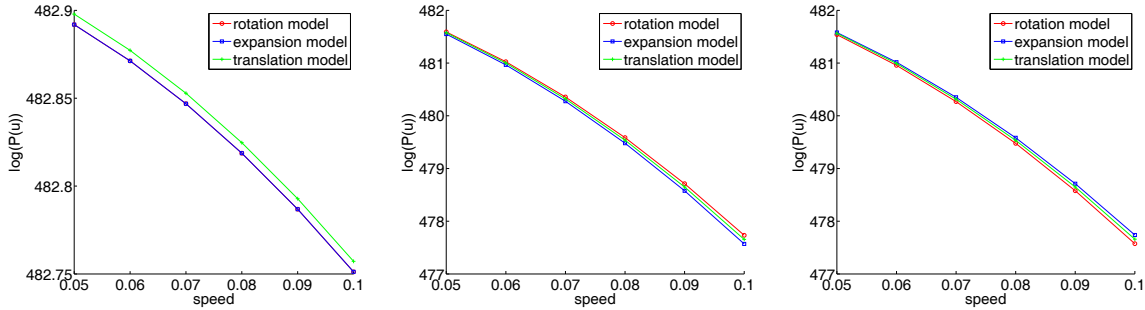

Figure 3: Model selection results with random dot motion. Plots the log probability of the model as a function of speed for each type of stimuli. left: translation stimuli; middle: rotation stimuli; right: expansion stimuli. Green curves with cross are from translation model. Red curves with circles are from rotation model. Blue curves with squares are from expansion model.

## 4.2   Speed threshold of Detection

As reported in [4], humans have lower speed threshold in detecting rotation/expansion than translation motion. The experiment is formulated as a model selection task with an additional "static" motion prior. The "static" motion prior is modeled as a translation prior with $\mu = 0$ and $\lambda$ significantly large to emphasize slowness. In the simulation, $\lambda = 0.3$ for this "static" model, while $\lambda = 0.001$ for all other models.

At low speed, the "static" model is favored due to its stronger bias towards slowness, as stimulus speed increases, it loses its advantage to other models. The speed thresholds of detection for different motion patterns can be seen from the model evidence plots in figure (4), and they are lower for rotation/expansion than translation. The threshold values are about 0.05 for rotation and expansion and 0.1 for translation. This is consistent with experimental result in [4].

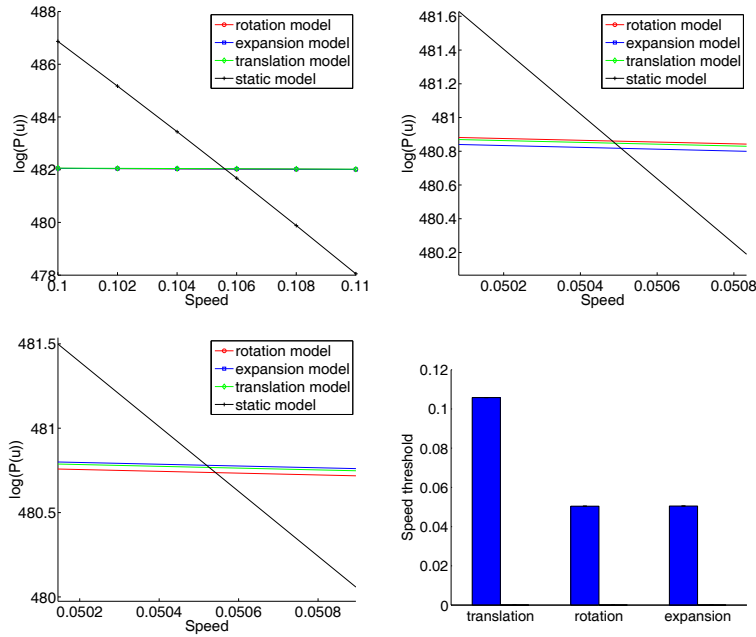

Figure 4: Speed threshold of detection. Upper left panel: model evidence plot for translation stimuli. Upper right panel: model evidence plot for rotation stimuli. Lower left panel: model eviddence plot for expansion stimuli. Lower right panel: bar graph of speed thresholds.

# 5  Results on randomly oriented gratings

## 5.1  Stimuli

When randomly oriented grating elements drift behind apertures, the perceived direction of motion is heavily biased by the orientation of the gratings, as well as by the shape and contrast of the apertures. Recently, Nishida and his colleagues developed a novel global motion stimulus consisting of a number of gratings elements, each with randomly assigned orientation [10]. A coherent motion is perceived when the drifting velocities of all elements are consistent with a given velocity. Examples of the stimuli used in these psychophysical experiments are shown in left side of figure (6). The stimuli consisted of 728 gratings (drifting sine-wave gratings windowed by stationary Gaussians). The orientations of the gratings were randomly assigned, and their drifting velocities were determined by a specified global motion flow pattern. The motions of signal grating elements were consistent with global motion, but the motions of noise grating elements were randomized. The task was to identify the global motion direction as one of two alternatives: left/right for translation, clockwise/counterclockwise for rotation, and inward/outward for expansion. Motion sensitivity was measured by the coherence threshold, defined as the proportion of signal elements that yielded a performance level of 75% correct.

Similar stimuli with 328 gratings were generated to test our computational models. The input for the models is the velocity component perpendicular to the assigned orientation for each grating, as illustrated in the upper two panels of figure (5).

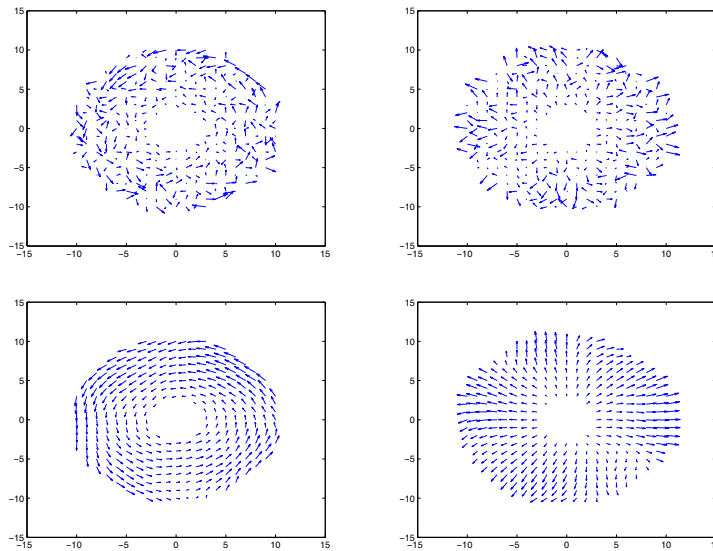

Figure 5: Randomly-oriented grating stimuli and estimated motion flow. Upper left panel: rotation stimulus (with 75% coherence ratio). Upper right panel: expansion stimulus (with 75% coherence ratio). Lower left panel: motion flow estimated from stimulus in first panel with rotation model. Lower right panel: motion flow estimated from stimulus in second panel with expansion model.

## 5.2  Result

The results of psychophysical experiments (middle panel of figure 6) showed worse performance for perceiving translation than rotation/expansion motion [6]. Clearly, as shown in the third panel of the same figure, the model performs best for rotation and expansion, and is worst for translation. This finding agrees with human performance in psychophysical experiments.

# 6  Conclusion

Humans motion sensitivities depend on the motion patterns (translation/rotation/expansion). We propose a computational model in which different prior motions compete to fit the data by levels

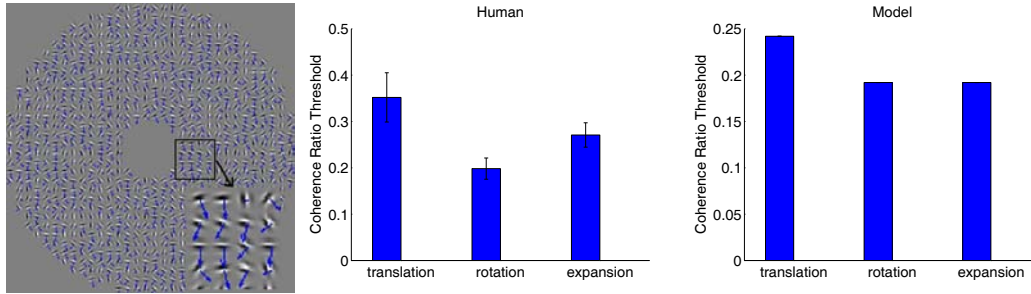

Figure 6: Stimulus and results. Left panel: illustration of grating stimulus. Blue arrows indicate the drifting velocity of each grating. Middle panel: human coherence thresholds for different motion stimuli. Right panel: Model prediction of coherence thresholds which are consistent with human trends.

of inference. This analysis involves formulating two new prior models for rotation and expansion model and deriving their properties. This *competitive prior* approach gives good fits to the empirical data and accounts for the dominant trends reported in [4, 6].

Our current work aims to extend these findings to a range of different motions (e.g. affine motion) and to use increasingly naturalistic appearance/intensity models. It is also important to determine if motion patterns to which humans are sensitive correspond to those appearing regularly in natural motion sequences.

# References

[1] J.F. Barraza and N.M. Grzywacz. Measurement of angular velocity in the perception of rotation. Vision Research, 42.2002.

[2] J. Duchon. Lecture Notes in Math. 571, (eds Schempp, W. and Zeller, K.) 85-100. Springer-Verlag, Berlin, 1979.

[3] C. J. Duffy, and R. H. Wurtz. Sensitivity of MST neurons to optic flow stimuli. I. A continuum of response selectivity to large field stimuli. Journal of Neurophysiology. 65, 1329-1345. 1991.

[4] T. Freeman, and M. Harris. Human sensitivity to expanding and rotating motion: effect of complementary masking and directional structure. Vision research, 32, 1992.

[5] D. Knill and W. Richards (Eds). Perception as Bayesian Inference. Cambridge University Press, 1996.

[6] A. Lee, A. Yuille, and H. Lu. Superior perception of circular/radial than translational motion cannot be explained by generic priors. VSS 2008.

[7] H. Lu and A.L. Yuille. Ideal Observers for Detecting Motion: Correspondence Noise. NIPS 2005.

[8] M. C. Morrone, D. C. Burr, and L. Vaina. Two stages of visual processing for radial and circular motion. Nature, 376, 507-509. 1995.

[9] M. Morrone, M. Tosetti, D. Montanaro, A. Fiorentini, G. Cioni, and D. C. Burr. A cortical area that responds specifically to optic flow revealed by fMRI. Nature Neuroscience, 3, 1322 -1328. 2000.

[10] S. Nishida, K. Amano, M. Edwards, and D.R. Badcock. Global motion with multiple Gabors - A tool to investigate motion integration across orientation and space. VSS 2006.

[11] R. Sekuler, S.N.J. Watamaniuk and R. Blake. Perception of Visual Motion. In Steven's Handbook of Experimental Psychology. Third edition. H. Pashler, series editor. S. Yantis, volume editor. J. Wiley Publishers. New York. 2002.

[12] A.A. Stocker and E.P. Simoncelli. Noise characteristics and prior expectations in human visual speed perception Nature Neuroscience, vol. 9(4), pp. 578–585, Apr 2006.

[13] A.A. Stocker, and E. Simoncelli. A Bayesian model of conditioned perception. *Proceedings of Neural Information Processing Systems*. 2007.

[14] K. Tanaka, Y. Fukada, and H. Saito. Underlying mechanisms of the response specificity of expansion/contraction and rotation cells in the dorsal part of the MST area of the macaque monkey. Journal of Neurophysiology. 62, 642-656. 1989.

[15] Y. Weiss, and E.H. Adelson. Slow and smooth: A Bayesian theory for the combination of local motion signals in human vision Technical Report 1624. Massachusetts Institute of Technology. 1998.

[16] Y. Weiss, E.P. Simoncelli, and E.H. Adelson. Motion illusions as optimal percepts. Nature Neuroscience, 5, 598-604. 2002.

[17] A.L. Yuille and N.M. Grzywacz. A computational theory for the perception of coherent visual motion. *Nature*, 333,71-74. 1988.
